# Extending Q-Learning to General Adaptive Multi-Agent Systems

**Gerald Tesauro**
IBM Thomas J. Watson Research Center
19 Skyline Drive, Hawthorne, NY 10532 USA
tesauro@watson.ibm.com

## Abstract

Recent multi-agent extensions of Q-Learning require knowledge of other agents' payoffs and Q-functions, and assume game-theoretic play at all times by all other agents. This paper proposes a fundamentally different approach, dubbed "Hyper-Q" Learning, in which values of mixed strategies rather than base actions are learned, and in which other agents' strategies are estimated from observed actions via Bayesian inference. Hyper-Q may be effective against many different types of adaptive agents, even if they are persistently dynamic. Against certain broad categories of adaptation, it is argued that Hyper-Q may converge to exact optimal time-varying policies. In tests using Rock-Paper-Scissors, Hyper-Q learns to significantly exploit an Infinitesimal Gradient Ascent (IGA) player, as well as a Policy Hill Climber (PHC) player. Preliminary analysis of Hyper-Q against itself is also presented.

## 1 Introduction

The question of how agents may adapt their strategic behavior while interacting with other arbitrarily adapting agents is a major challenge in both machine learning and multi-agent systems research. While game theory provides a pricipled calculation of Nash equilibrium strategies, it is limited in practical use due to hidden or imperfect state information, and computational intractability. Trial-and-error learning could develop good strategies by trying many actions in a number of environmental states, and observing which actions, in combination with actions of other agents, lead to high cumulative reward. This is highly effective for a single learner in a stationary environment, where algorithms such as Q-Learning [13] are able to learn optimal policies on-line without a model of the environment. Straight off-the-shelf use of RL algorithms such as Q-learning is problematic, however, because: (a) they learn deterministic policies, whereas mixed strategies are generally needed; (b) the environment is generally non-stationary due to adaptation of other agents.

Several multi-agent extensions of Q-Learning have recently been published. Littman [7] developed a convergent algorithm for two-player zero-sum games. Hu and Wellman [5] present an algorithm for two-player general-sum games, the convergence of which was clarified by Bowling [1]. Littman [8] also developed a convergent many-agent "friend-or-foe" Q-learning algorithm combining cooperative learning with adversarial learning. These all extend the normal Q-function of state-action pairs $Q(s, a)$ to a function of states and joint actions of all agents, $Q(s, \vec{a})$. These algorithms make a number of strong assumptions

which facilitate convergence proofs, but which may not be realistic in practice. These include: (1) other agents' payoffs are fully observable; (2) all agents use the same learning algorithm; (3) during learning, other agents' strategies are derivable via game-theoretic analysis of the current Q-functions. In particular, if the other agents employ non-game-theoretic or nonstationary strategies, the learned Q-functions will not accurately represent the expected payoffs obtained by playing against such agents, and the associated greedy policies will not correspond to best-reponse play against the other agents.

The aim of this paper is to develop more general and practical extensions of Q-learning avoiding the above assumptions. The multi-agent environment is modeled as a repeated stochastic game in which other agents' actions are observable, but not their payoffs. Other agents are assumed to learn, but the forms of their learning algorithms are unknown, and their strategies may be asymptotically non-stationary. During learning, it is proposed to estimate other agents' current strategies from observation instead of game-theoretic analysis.

The above considerations lead to a new algorithm, presented in Section 2 of the paper, called "Hyper-Q Learning." Its key idea is to learn the value of joint mixed strategies, rather than joint base actions. Section 3 discusses the effects of function approximation, exploration, and other agents' strategy dynamics on Hyper-Q's convergence. Section 4 presents a Bayesian inference method for estimating other agents' strategies, by applying a recency-weighted version of Bayes' rule to the observed action sequence. Section 5 discusses implementation details of Hyper-Q in a simple Rock-Paper-Scissors test domain. Test results are presented against two recent algorithms for learning mixed strategies: Infinitesimal Gradient Ascent (IGA) [10], and Policy Hill Climbing (PHC) [2]. Preliminary results of Hyper-Q vs. itself are also discussed. Concluding remarks are given in section 6.

## 2 General Hyper-Q formulation

An agent using normal Q-learning in a finite MDP repeatedly observes a state $s$, chooses a legal action $a$, and then observes an immediate reward $r$ and a transition to a new state $s'$. The Q-learning equation is given by: $\Delta Q(s, a) = \alpha(t)[r + \gamma \max_b Q(s', b) - Q(s, a)]$, where $\gamma$ is a discount parameter, and $\alpha(t)$ is an appropriate learning rate schedule. Given a suitable method of exploring state-action pairs, Q-learning is guaranteed to converge to the optimal value function $Q^*$, and its associated greedy policy is thus an optimal policy $\pi^*$.

The multi-agent generalization of an MDP is called a *stochastic game*, in which each agent $i$ chooses an action $a_i$ in state $s$. Payoffs $r_i$ to agent $i$ and state transitions are now functions of joint actions of all agents. An important special class of stochastic games are matrix games, in which $|S| = 1$ and payoffs are functions only of joint actions. Rather than choosing the best action in a given state, an agent's task in a stochastic game is to choose the best *mixed strategy* $\vec{x}_i = \vec{x}_i(s)$ given the expected mixed strategy $\vec{x}_{-i}(s)$ of all other agents. Here $\vec{x}_i$ denotes a set a probabilities summing to 1 for selecting each of the $N_i = N_i(s)$ legal actions in state $s$. The space of possible mixed strategies is a continuous $(N_i - 1)$ dimensional unit simplex, and choosing the best mixed strategy is clearly more complex than choosing the best base action.

We now consider extensions of Q-learning to stochastic games. Given that the agent needs to learn a mixed strategy, which may depend on the mixed strategies of other agents, an obvious idea is to have the Q-function evaluate entire mixed strategies, rather than base actions, and to include in the "state" description an observation or estimate of the other agents' current mixed strategy. This forms the basis of the proposed Hyper-Q learning algorithm, which is formulated as follows. For notational simplicity, let $x$ denote the Hyper-Q learner's current mixed strategy, and let $y$ denote an estimated joint mixed strategy of all other agents (hereafter referred to as "opponents"). At time $t$, the agent generates a base action according to $x$, and then observes a payoff $r$, a new state $s'$, and a new estimated opponent strategy $y'$. The Hyper-Q function $Q(s, y, x)$ is then adjusted according to:

$$\Delta Q(s, y, x) = \alpha(t)[r + \gamma \max_{x'} Q(s', y', x') - Q(s, y, x)] \tag{1}$$

The greedy policy $\hat{x}$ associated with any Hyper-Q function is then defined by:

$$\hat{x}(s, y) = \arg \max_{x} Q(s, y, x) \tag{2}$$

## 3  Convergence of Hyper-Q Learning

### 3.1  Function approximation

Since Hyper-Q is a function of continuous mixed strategies, one would expect it to require some sort of function approximation scheme. Establishing convergence of Q-learning with function approximation is substantially more difficult than for a normal Q-table for a finite MDP, and there are a number of well-known counterexamples. In particular, finite discretization may cause a loss of an MDP's Markov property [9].

Several recent function approximation schemes [11, 12] enable Q-learning to work well in continuous spaces. There is a least one discretization scheme, *Finite Difference Reinforcement Learning* [9], that provably converges to the optimal value function of the underlying continuous MDP. This paper employs a simple uniform grid discretization of the mixed strategies of the Hyper-Q agent and its opponents. No attempt will be made to prove convergence under this scheme. However, for certain types of opponent dynamics described below, a plausible conjecture is that a Finite-Difference-RL implementation of Hyper-Q will be provably convergent.

### 3.2  Exploration

Convergence of normal Q-learning requires visiting every state-action pair infinitely often. The clearest way to achieve this in simulation is via *exploring starts*, in which training consists of many episodes, each starting from a randomly selected state-action pair. For real environments where this may not be feasible, one may utilize off-policy randomized exploration, e.g., $\epsilon$-greedy policies. This will ensure that, for all visited states, every action will be tried infinitely often, but does not guarantee that all states will be visited infinitely often (unless the MDP has an ergodicity property). As a result one would not expect the trained Q function to exactly match the ideal optimal $Q^*$ for the MDP, although the difference in expected payoffs of the respective policies should be vanishingly small.

The above considerations should apply equally to Hyper-Q learning. The use of exploring starts for states, agent and opponent mixed strategies should guarantee sufficient exploration of the state-action space. Without exploring starts, the agent can use $\epsilon$-greedy exploration to at least obtain sufficient exploration of its own mixed strategy space. If the opponents also do similar exploration, the situation should be equivalent to normal Q-learning, where some stochastic game states might not be visited infinitely often, but the cost in expected payoff should be vanishingly small. If the opponents do not explore, the effect could be a further reduction in effective state space explored by the Hyper-Q agent (where "effective state" = stochastic game state plus opponent strategy state). Again this should have a negligible effect on the agent's long-run expected payoff relative to the policy that would have been learned with opponent exploration.

### 3.3  Opponent strategy dynamics

Since opponent strategies can be governed by arbitrarily complicated dynamical rules, it seems unlikely that Hyper-Q learning will converge for arbitrary opponents. Nevertheless, some broad categories can be identified under which convergence should be achievable. One simple example is that of a stationary opponent strategy, i.e., $y(s)$ is a constant. In this

case, the stochastic game obviously reduces to an equivalent MDP with stationary state transitions and stationary payoffs, and with the appropriate conditions on exploration and learning rates, Hyper-Q will clearly converge to the optimal value function.

Another important broad class of dynamics consists of opponent strategies that evolve according to a fixed, history-independent rule depending only on themselves and not on actions of the Hyper-Q player, i.e., $y_{t+1} = f(s, y_t)$. This is a reasonable approximation for many-player games in which any individual has negligible "market impact," or in which a player's influence on another player occurs only through a global summarization function [6]. In such cases the relevant population strategy representation need only express global summarizations of actitivy (e.g. averages), not details of which player does what. An example is the "Replicator Dynamics" model from evolutionary game theory [14], in which a strategy grows or decays in a population according to its fitness relative to the population average fitness. This leads to a history independent first order differential equation $\dot{y} = f(y)$ for the population average strategy. In such models, the Hyper-Q learner again faces an effective MDP in which the effective state $(s, y)$ undergoes stationary history-independent transitions, so that Hyper-Q should be able to converge.

A final interesting class of dynamics occurs when the opponent can accurately estimate the Hyper-Q strategy $x$, and then adapts its strategy using a fixed history-independent rule: $y_{t+1} = f(s, y_t, x_t)$. This can occur if players are required to announce their mixed strategies, or if the Hyper-Q player voluntarily announces its strategy. An example is the Infinitesimal Gradient Ascent (IGA) model [10], in which the agent uses knowledge of the current strategy pair $(x, y)$ to make a small change in its strategy in the direction of the gradient of immediate payoff $P(x, y)$. Once again, this type of model reduces to an MDP with stationary history-independent transitions of effective state depending only on $(s, y, x)$.

Note that the above claims of reduction to an MDP depend on the Hyper-Q learner being able to accurately estimate the opponent mixed strategy $y$. Otherwise, the Hyper-Q learner would face a POMDP situation, and standard convergence proofs would not apply.

## 4 Opponent strategy estimation

We now consider estimation of opponent strategies from the history of base actions. One approach to this is *model-based*, i.e., to consider a class of explicit dynamical models of opponent strategy, and choose the model that best fits the observed data. There are two difficult aspects to this approach: (1) the class of possible dynamical models may need to be extraordinarily large; (2) there is a well-known danger of "infinite regress" of opponent models if A's model of B attempts to take into account B's model of A.

An alternative approach studied here is *model-free* strategy estimation. This is in keeping with the spirit of Q-learning, which learns state valuations without explicitly modeling the dynamics of the underlying state transitions. One simple method used in the following section is the well-known Exponential Moving Average (EMA) technique. This maintains a moving average $\bar{y}$ of opponent strategy by updating after each observed action using:

$$\bar{y}(t+1) = (1 - \mu)\bar{y}(t) + \mu\vec{u}_a(t) \tag{3}$$

where $\vec{u}_a(t)$ is a unit vector representation of the base action $a$. EMA assumes only that recent observations are more informative than older observations, and should give accurate estimates when significant strategy changes take place on time scales $> O(1/\mu)$.

### 4.1 Bayesian strategy estimation

A more principled model-free alternative to EMA is now presented. We assume a discrete set of possible values of $y$ (e.g. a uniform grid). A probability for each $y$ given the history of observed actions $H$, $P(y|H)$, can then be computed using Bayes' rule as follows:

$$P(y|H) = \frac{P(H|y)P(y)}{\sum_{y'} P(H|y')P(y')} \qquad (4)$$

where $P(y)$ is the prior probability of state $y$, and the sum over $y'$ extends over all strategy grid points. The conditional probability of the history given the strategy, $P(H|y)$, can now be decomposed into a product of individual action probabilities $\prod_{k=0}^{t} P(a(k)|y(t))$ assuming conditional independence of the individual actions. If all actions in the history are equally informative regardless of age, we may write $P(a(k)|y(t)) = y_{a(k)}(t)$ for all $k$. This corresponds to a Naive-Bayes equal weighting of all observed actions. However, it is again reasonable to assume that more recent actions are more informative. The way to implement this in a Bayesian context is with exponent weights $w_k$ that increase with $k$ [4]. Within a normalization factor, we then write:

$$P(H|y) = \prod_{k=0}^{t} y_{a(k)}^{w_k} \qquad (5)$$

A linear schedule $w_k = 1 - \mu(t - k)$ for the weights is intuitively obvious; truncation of the history at the most recent $1/\mu$ observations ensures that all weights are positive.

## 5  Implementation and Results

We now examine the performance of Hyper-Q learning in a simple two-player matrix game, Rock-Paper-Scissors. A uniform grid discretization of size $N = 25$ is used to represent mixed-strategy component probabilities, giving a simplex grid of size $N(N + 1)/2 = 325$ for either player's mixed strategy, and thus the entire Hyper-Q table is of size $(325)^2 = 105625$. All simulations use $\gamma = 0.9$, and for simplicity, a constant learning rate $\alpha = 0.01$.

### 5.1  Hyper-Q/Bayes formulation

Three different opponent estimation schemes were used with Hyper-Q learning: (1) "Omniscient," i.e. perfect knowledge of the opponent's strategy; (2) EMA, using equation 3 with $\mu = 0.005$; (3) Bayesian, using equations 4 and 5 with $\mu = 0.005$ and a uniform prior. Equations 1 and 2 were modified in the Bayesian case to allow for a distribution of opponent states $y$, with probabilities $P(y|H)$. The corresponding equations are:

$$\Delta Q(y, x) = \alpha(t)P(y|H)[r + \gamma \max_{x'} Q(y', x') - Q(y, x)] \qquad (6)$$

$$\hat{x} = \arg\max_x \sum_y P(y|H)Q(y, x) \qquad (7)$$

A technical note regarding equation 6 is that, to improve tractability of the algorithm, an approximation $P(y|H) \approx P(y'|H')$ is used, so that the Hyper-Q table updates are performed using the updated distribution $P(y'|H')$.

### 5.2  Rock-Paper-Scissors results

We first examine Hyper-Q training online against an IGA player. Apart from possible state observability and discretization issues, Hyper-Q should in principle be able to converge against this type of opponent. In order to conform to the original implicit assumptions underlying IGA, the IGA player is allowed to have omniscient knowledge of the Hyper-Q player's mixed strategy at each time step. Policies used by both players are always greedy, apart from resets to uniform random values every 1000 time steps.

Figure 1 shows a smoothed plot of the online Bellman error, and the Hyper-Q player's average reward per time step, as a function of training time. The figure exhibits good

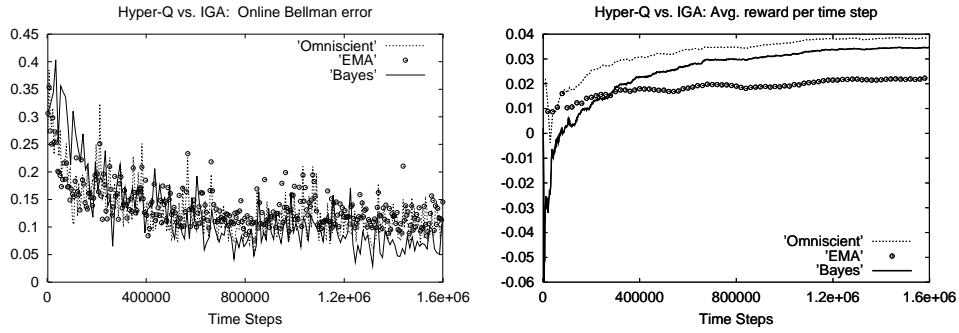

Figure 1: Results of Hyper-Q learning vs. an IGA player in Rock-Paper-Scissors, using three different opponent state estimation methods: "Omniscient," "EMA" and "Bayes" as indicated. Random strategy restarts occur every 1000 time steps. Left plot shows smoothed online Bellman error. Right plot shows average Hyper-Q reward per time step.

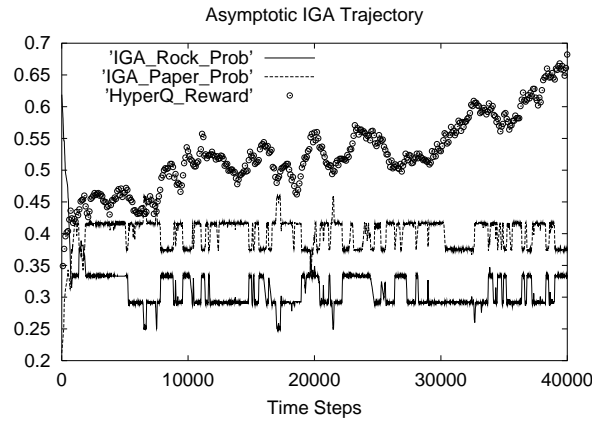

Figure 2: Trajectory of the IGA mixed strategy against the Hyper-Q strategy starting from a single exploring start. Dots show Hyper-Q player's cumulative (rescaled) reward.

progress toward convergence, as suggested by substantially reduced Bellman error and substantial positive average reward per time step. Among the three estimation methods used, Bayes reached the lowest Bellman error at long time scales. This is probably because it updates many elements in the Hyper-Q table per time step, whereas the other techniques only update a single element. Bayes also has by far the worst average reward at the start of learning, but asymptotically it clearly outperforms EMA, and comes close to matching the performance obtained with omniscient knowledge of opponent state.

Part of Hyper-Q's advantage comes from exploiting transient behavior starting from a random initial condition. In addition, Hyper-Q also exploits the asymptotic behavior of IGA, as shown in figure 2. This plot shows that the initial transient lasts at most a few thousand time steps. Afterwards, the Hyper-Q policy causes IGA to cycle erratically between two different probabilites for Rock and two different probabilities for Paper, thus preventing IGA from reaching the Nash mixed strategy. The overall profit to Hyper-Q during this cycling is positive on average, as shown by rising cumulative Hyper-Q reward. The observed cycling with positive profitability is reminiscent of an algorithm called PHC-Exploiter [3] in play against a PHC player. An interesting difference is that PHC-Exploiter uses an explicit model of its opponent's behavior, whereas no such model is needed by a Hyper-Q learner.

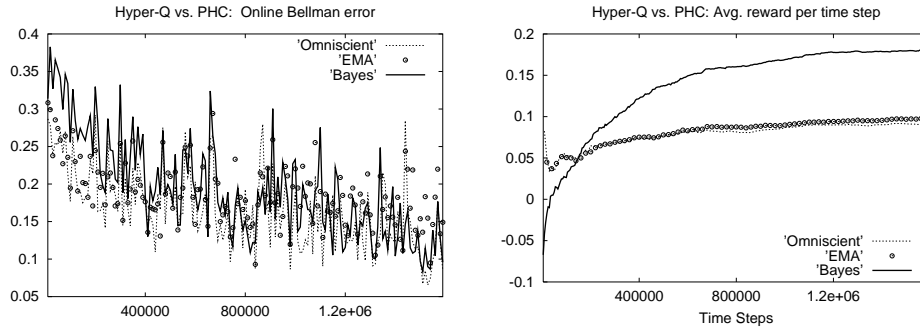

Figure 3: Results of Hyper-Q vs. PHC in Rock-Paper-Scissors. Left plot shows smoothed online Bellman error. Right plot shows average Hyper-Q reward per time step.

We now exmamine Hyper-Q vs. a PHC player. PHC is a simple adaptive strategy based only on its own actions and rewards. It maintains a Q-table of values for each of its base actions, and at every time step, it adjusts its mixed strategy by a small step towards the greedy policy of its current Q-function. The PHC strategy is history-dependent, so that reduction to an MDP is not possible for the Hyper-Q learner. Nevertheless Hyper-Q does exhibit substantial reduction in Bellman error, and also significantly exploits PHC in terms of average reward, as shown in figure 3. Given that PHC ignores opponent state, it should be a weak competitive player, and in fact it does much worse in average reward than IGA. It is also interesting to note that Bayesian estimation once again clearly outperforms EMA estimation, and surprisingly, it also outperforms omniscient state knowledge. This is not yet understood and is a focus of ongoing research.

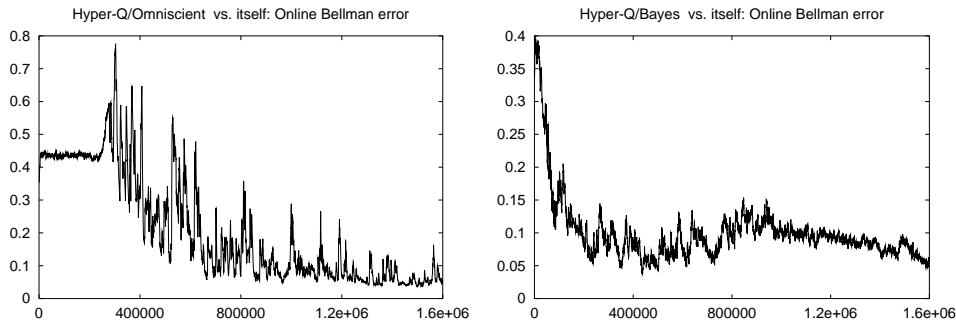

Figure 4: Smoothed online Bellman error for Hyper-Q vs. itself. Left plot uses Omniscient state estimation; right plot uses Bayesian estimation.

Finally, we examine preliminary data for Hyper-Q vs. itself. The average reward plots are uninteresting: as one would expect, each player's average reward is close to zero. The online Bellman error, shown in figure 4, is more interesting. Surprisingly, the plots are less noisy and achieve asymptotic errors as low or lower than against either IGA or PHC. Since Hyper-Q's play is history-dependent, one can't argue for MDP equivalence. However, it is possible that the players' greedy policies $\hat{x}(y)$ and $\hat{y}(x)$ simultaneously become stationary, thereby enabling them to optimize against each other. In examining the actual play, it does not converge to the Nash point $(\frac{1}{3}, \frac{1}{3}, \frac{1}{3})$, but it does appear to cycle amongst a small number of grid points with roughly zero average reward over the cycle for both players. Conceivably, Hyper-Q could have converged to a cyclic Nash equilibrium of the repeated game, which would certainly be a nice outcome of self-play learning in a repeated game.

# 6 Conclusion

Hyper-Q Learning appears to be more versatile and general-purpose than any published multi-agent extension of Q-Learning to date. With grid discretization it scales badly but with other function approximators it may become practical. Some tantalizing early results were found in Rock-Paper-Scissors tests against some recently published adaptive opponents, and also against itself. Research on this topic is very much a work in progress. Vastly more research is needed, to develop a satisfactory theoretical analysis of the approach, an understanding of what kinds of realistic environments it can be expcted to do well in, and versions of the algorithm that can be successfully deployed in those environments.

Significant improvements in opponent state estimation should be easy to obtain. More principled methods for setting recency weights should be achievable; for example, [4] proposes a method for training optimal weight values based on observed data. The use of time-series prediction and data mining methods might also result in substantially better estimators. Model-based estimators are also likely to be advantageous where one has a reasonable basis for modeling the opponents' dynamical behavior.

**Acknowledgements:** The author thanks Michael Littman for many helpful discussions; Irina Rish for insights into Bayesian state estimation; and Michael Bowling for assistance in implementing the PHC algorithm.

# References

[1] M. Bowling. Convergence problems of general-sum multiagent reinforcement learning. In *Proceedings of ICML-00*, pages 89–94, 2000.

[2] M. Bowling and M. Veloso. Multiagent learning using a variable learning rate. *Artificial Intelligence*, 136:215–250, 2002.

[3] Y.-H. Chang and L. P. Kaelbling. Playing is believing: the role of beliefs in multi-agent learning. In *Proceedings of NIPS-2001*. MIT Press, 2002.

[4] S. J. Hong, J. Hosking, and R. Natarajan. Multiplicative adjustment of class probability: educating naive Bayes. Technical Report RC-22393, IBM Research, 2002.

[5] J. Hu and M. P. Wellman. Multiagent reinforcement learning: theoretical framework and an algorithm. In *Proceedings of ICML-98*, pages 242–250. Morgan Kaufmann, 1998.

[6] M. Kearns and Y. Mansour. Efficient Nash computation in large population games with bounded influence. In *Proceedings of UAI-02*, pages 259–266, 2002.

[7] M. L. Littman. Markov games as a framework for multi-agent reinforcement learning. In *Proceedings of ICML-94*, pages 157–163. Morgan Kaufmann, 1994.

[8] M. L. Littman. Friend-or-Foe Q-learning in general-sum games. In *Proceedings of ICML-01*. Morgan Kaufmann, 2001.

[9] R. Munos. A convergent reinforcement learning algorithm in the continuous case based on a finite difference method. In *Proceedings of IJCAI-97*, pages 826–831. Morgan Kaufman, 1997.

[10] S. Singh, M. Kearns, and Y. Mansour. Nash convergence of gradient dynamics in general-sum games. In *Proceedings of UAI-2000*, pages 541–548. Morgan Kaufman, 2000.

[11] W. D. Smart and L. P. Kaelbling. Practical reinforcement learning in continuous spaces. In *Proceedings of ICML-00*, pages 903–910, 2000.

[12] W. T. B. Uther and M. M. Veloso. Tree based discretization for continuous state space reinforcement learning. In *Proceedings of AAAI-98*, pages 769–774, 1998.

[13] C. Watkins. *Learning from Delayed Rewards*. PhD thesis, Cambridge University, 1989.

[14] J. W. Weibull. *Evolutionary Game Theory*. The MIT Press, 1995.
